# Bayesian Model Scoring in Markov Random Fields

**Sridevi Parise**
Bren School of Information and Computer Science
UC Irvine
Irvine, CA 92697-3425
sparise@ics.uci.edu

**Max Welling**
Bren School of Information and Computer Science
UC Irvine
Irvine, CA 92697-3425
welling@ics.uci.edu

## Abstract

Scoring structures of undirected graphical models by means of evaluating the marginal likelihood is very hard. The main reason is the presence of the partition function which is intractable to evaluate, let alone integrate over. We propose to approximate the marginal likelihood by employing two levels of approximation: we assume normality of the posterior (the Laplace approximation) and approximate all remaining intractable quantities using belief propagation and the linear response approximation. This results in a fast procedure for model scoring. Empirically, we find that our procedure has about two orders of magnitude better accuracy than standard BIC methods for small datasets, but deteriorates when the size of the dataset grows.

## 1 Introduction

Bayesian approaches have become an important modeling paradigm in machine learning. They offer a very natural setting in which to address issues such as overfitting which plague standard maximum likelihood approaches. A full Bayesian approach has its computational challenges as it often involves intractable integrals. While for Bayesian networks many of these challenges have been met successfully[3], the situation is quite reverse for Markov random field models. In fact, it is very hard to find any literature at all on model order selection in general MRF models. The main reason for this discrepancy is the fact that MRF models have a normalization constant that depends on the parameters but is in itself intractable to compute, let alone integrate over. In fact, the presence of this term even prevents one to draw samples from the posterior distribution in most situations except for some special cases[1].

In terms of approximating the posterior some new methods have become available recently. In [7] a number of approximate MCMC samplers are proposed. Two of them were reported to be most successful: one based on Langevin sampling with approximate gradients given by contrastive divergence and one where the acceptance probability is approximated by replacing the log partition function with the Bethe free energy. Both these methods are very general, but inefficient. In [2] MCMC methods are explored for the Potts model based on the reversible jump formalism. To compute acceptance ratios for dimension-changing moves they need to estimate the partition function

using a separate estimation procedure making it rather inefficient as well. In [6] and [8] MCMC methods are proposed that use perfect samples to circumvent the calculation of the partition function altogether. This method is elegant but limited in its application due to the need to draw perfect samples. Moreover, two approaches that approximate the posterior by a Gaussian distribution are proposed in [11] (based on expectation propagation) and [13] (based on the Bethe-Laplace approximation).

In this paper we focus on a different problem, namely that of approximating the marginal likelihood. This quantity is at the heart of Bayesian analysis because it allows one to compare models of different structure. One can use it to either optimize or average over model structures. Even if one has an approximation to the posterior distribution it is not at all obvious how to use it to compute a good estimate for the marginal likelihood. The most direct approach is to use samples from the posterior and compute importance weights,

$$ p(D) \approx \frac{1}{N} \sum_{n=1}^{N} p(D|\theta_n)p(\theta_n)/Q(\theta_n|D) \qquad \theta_n \sim Q(\theta_n|D) \tag{1} $$

where $Q(\theta_n|D)$ denotes the approximate posterior. Unfortunately, this importance sampler suffers from very high variance when the number of parameters becomes large. It is not untypical that the estimate is effectively based on a single example.

We propose to use the Laplace approximation, including all $\mathcal{O}(1)$ terms where the intractable quantities of interest are approximated by either belief propagation (BP) or the linear response theorem based on the solution of BP. We show empirically that the $\mathcal{O}(1)$ terms are indispensable for small $N$. Their inclusion can improve accuracy to up to two orders of magnitude. At the same time we observe that as a function of $N$, the $\mathcal{O}(1)$-term based on the covariance between features deteriorates and should be omitted for large $N$. We conjecture that this phenomenon is explained by the fact that the calculation of the covariance between features, which is equal to the second derivative of the log-normalization constant, becomes instable if the bias in the MAP estimate of the parameters is of the order of the variance in the posterior. For any biased estimate of the parameters this phenomenon is therefore bound to happen as we increase $N$ because the variance of the posterior distribution is expected to decrease with $N$.

In summary we present a very accurate estimate for the marginal likelihood where it is most needed, i.e. for small $N$. This work seems to be the first practical method for estimating the marginal evidence in undirected graphical models.

## 2   The Bethe-Laplace Approximation for $\log p(D)$

Without loss of generality we represent a MRF as a log-linear model,

$$ p(\mathbf{x}|\boldsymbol{\lambda}) = \frac{1}{Z(\boldsymbol{\lambda})} \exp\left[ \boldsymbol{\lambda}^T \mathbf{f}(\mathbf{x}) \right] \tag{2} $$

where $\mathbf{f}(\mathbf{x})$ represent features. In the following we will assume that the random variables $\mathbf{x}$ are observed. Generalizations to models with hidden variables exist in theory but we defer the empirical evaluation of this case to future research.

To score a structure we will follow the Bayesian paradigm and aim to compute the log-marginal likelihood $\log p(D)$ where $D$ represents a dataset of size $N$,

$$ \log p(D) = \log \int \mathrm{d}\boldsymbol{\lambda}\, p(D|\boldsymbol{\lambda})\, p(\boldsymbol{\lambda}) \tag{3} $$

where $p(\boldsymbol{\lambda})$ is some arbitrary prior on the parameters $\boldsymbol{\lambda}$.

In order to approximate this quantity we employ two approximations. Firstly, we expand the both log-likelihood and log-prior around the MAP value $\boldsymbol{\lambda}^{\text{MP}}$. For the log-likelihood this boils down to expanding the log-partition function,

$$ \log Z(\boldsymbol{\lambda}) \approx \log Z(\boldsymbol{\lambda}^{\text{MP}}) + \boldsymbol{\kappa}^T \delta\boldsymbol{\lambda} + \frac{1}{2}\delta\boldsymbol{\lambda}^T C \delta\boldsymbol{\lambda} \tag{4} $$

with $\delta\boldsymbol{\lambda} = (\boldsymbol{\lambda} - \boldsymbol{\lambda}^{\text{MP}})$ and

$$C = \mathbb{E}[\mathbf{f}(\mathbf{x})\mathbf{f}(\mathbf{x})^T]_{p(\mathbf{x})} - \mathbb{E}[\mathbf{f}(\mathbf{x})]_{p(\mathbf{x})}\mathbb{E}[\mathbf{f}(\mathbf{x})]_{p(\mathbf{x})}^T, \qquad \boldsymbol{\kappa} = \mathbb{E}[\mathbf{f}(\mathbf{x})]_{p(\mathbf{x})} \qquad (5)$$

and where all averages are taken over $p(\mathbf{x}|\boldsymbol{\lambda}^{\text{MP}})$.

Similarly for the prior we find,

$$\log p(\boldsymbol{\lambda}) = \log p(\boldsymbol{\lambda}^{\text{MP}}) + \mathbf{g}^T \delta\boldsymbol{\lambda} + \frac{1}{2}\delta\boldsymbol{\lambda}^T H \delta\boldsymbol{\lambda} \qquad (6)$$

where $\mathbf{g}$ is the first derivative of $\log p$ evaluated at $\boldsymbol{\lambda}^{\text{MP}}$ and $H$ is the second derivative (or Hessian).

The variables $\delta\boldsymbol{\lambda}$ represent fluctuations of the parameters around the MAP value $\boldsymbol{\lambda}^{\text{MP}}$. The marginal likelihood can now be approximated by integrating out the fluctuations $\delta\boldsymbol{\lambda}$, considering $\boldsymbol{\lambda}^{\text{MP}}$ as a hyper-parameter,

$$\log p(D) = \log \int d\delta\boldsymbol{\lambda}\, p(D|\delta\boldsymbol{\lambda}, \boldsymbol{\lambda}^{\text{MP}})\, p(\delta\boldsymbol{\lambda}|\boldsymbol{\lambda}^{\text{MP}}) \qquad (7)$$

Inserting the expansions eqns.4 and 6 into eqn.7 we arrive at the standard expression for the Laplace approximation applied to MRFs,

$$\log p(D) \approx \qquad\qquad\qquad\qquad\qquad\qquad\qquad\qquad\qquad\qquad\qquad\qquad (8)$$

$$\sum_n \boldsymbol{\lambda}^{\text{MP}\,T}\mathbf{f}(\mathbf{x}_n) - N \log Z(\boldsymbol{\lambda}^{\text{MP}}) + \log p(\boldsymbol{\lambda}^{\text{MP}}) + \frac{1}{2}F\log(2\pi) - \frac{1}{2}F\log(N) - \frac{1}{2}\log\det(C - \frac{H}{N})$$

with $F$ the number of features.

The difference with Laplace approximations for Bayesian networks is the fact that many terms in the expression above can not be evaluated. First of all, determining $\boldsymbol{\lambda}^{\text{MP}}$ requires running gradient ascent or iterative scaling to maximize the penalized log-likelihood which requires the computation of the average sufficient statistics $\mathbb{E}[\mathbf{f}(\mathbf{x})]_{p(\mathbf{x})}$. Secondly, the expression contains the log-partition function $Z(\boldsymbol{\lambda}^{\text{MP}})$ and the covariance matrix $C$ which are both intractable quantities.

## 2.1 The BP-Linear Response Approximation

To make further progress, we introduce a second layer of approximations based on belief propagation. In particular, we approximate the required marginals in the gradient for $\boldsymbol{\lambda}^{\text{MP}}$ with the ones obtained with BP. For fully observed MRFs the value for $\boldsymbol{\lambda}^{\text{MP}}$ will be very close to the solution obtained by pseudo-moment matching (PMM) [5]; the influence of the prior being the only difference between the two. Hence, we use $\boldsymbol{\lambda}^{\text{PMM}}$ to initialize gradient descent. The approximation incurred by PMM is not always small [10] in which case other approximations such as contrastive divergence may be substituted instead. The term $-\log Z(\boldsymbol{\lambda}^{\text{MP}})$ will be approximated with the Bethe free energy. This will involve running belief propagation on a model with parameters $\boldsymbol{\lambda}^{\text{MP}}$ and inserting the beliefs at their fixed points into the expression for the Bethe free energy [16].

To compute the covariance matrix between the features $C$ (eqn.5), we use the linear response algorithm of [15]. This approximation is based on the observation that $C$ is the Hessian of the log-partition function w.r.t. the parameters. This is approximated by the Hessian of the Bethe free energy w.r.t. the parameters which in turn depends to the partial derivatives of the beliefs from BP w.r.t. the parameters.

$$C_{\alpha\beta} = \frac{\partial^2 \log Z(\boldsymbol{\lambda})}{\partial\lambda_\alpha \partial\lambda_\beta} \approx -\frac{\partial^2 \log \mathcal{F}_{\text{Bethe}}(\boldsymbol{\lambda})}{\partial\lambda_\alpha \partial\lambda_\beta} = \sum_{x_\alpha} f_\alpha(x_\alpha)\frac{\partial p_\alpha^{\text{BP}}(x_\alpha|\boldsymbol{\lambda})}{\partial\lambda_\beta} \qquad (9)$$

where $\boldsymbol{\lambda} = \boldsymbol{\lambda}^{\text{MP}}$, $p_\alpha^{\text{BP}}$ is the marginal computed using belief propagation and $x_\alpha$ is the collection of variables in the argument of feature $f_\alpha$ (e.g. nodes or edges). This approximate $C$ is also guaranteed to be symmetric and positive semi-definite. In [15] two algorithms were discussed to compute $C$ in the linear response approximation, one based on a matrix inverse, the other a local propagation algorithm. The main idea is to perform a Taylor expansion of the beliefs and messages in the parameters $\delta\boldsymbol{\lambda} = \boldsymbol{\lambda} - \boldsymbol{\lambda}^{\text{MP}}$ and keep track of first order terms in the belief propagation equations. One can show that the first order terms carry the information to compute the covariance matrix. We refer to [15] for more information. In appendix A we provide explicit equations for the case of Boltzman machines which is what is needed to reproduce the experiments in section 4.

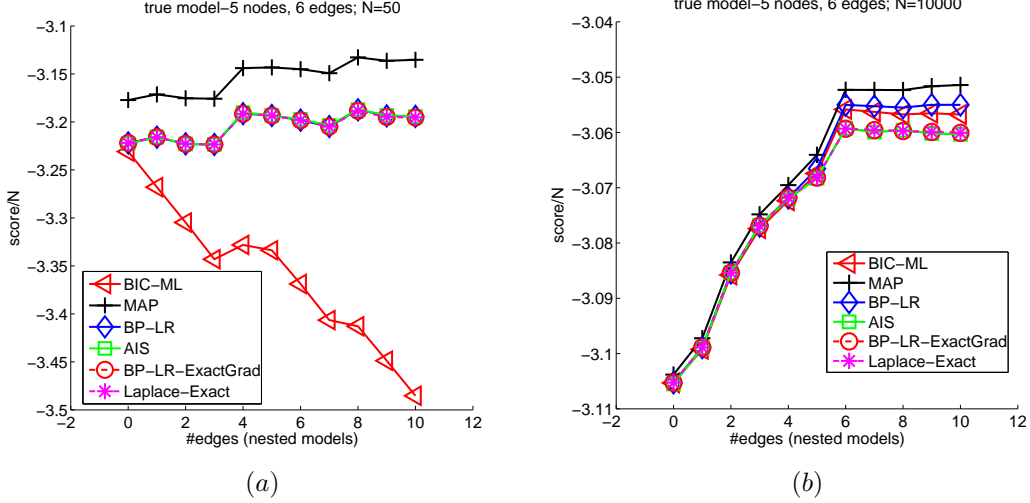

Figure 1: Comparision of various scores on synthetic data

## 3 Conditional Random Fields

Perhaps the most practical class of undirected graphical models are the conditional random field (CRF) models. Here we jointly model labels $\mathbf{t}$ and input variables $\mathbf{x}$. The most significant modification relative to MRFs is that the normalization term now depends on the input variable. The probability of label given input is given as,

$$p(\mathbf{t}|\mathbf{x}, \boldsymbol{\lambda}) = \frac{1}{Z(\boldsymbol{\lambda}, \mathbf{x})} \exp\left[\boldsymbol{\lambda}^T \mathbf{f}(\mathbf{t}, \mathbf{x})\right] \tag{10}$$

To approximate the log marginal evidence we obtain an expression very similar to eqn.8 with the following replacement,

$$\bullet \qquad \mathbf{C} \rightarrow \frac{1}{N} \sum_{n=1}^{N} C_{\mathbf{x}_n} \tag{11}$$

$$\bullet \qquad \sum_n \left(\boldsymbol{\lambda}^{\text{MP}T} \mathbf{f}(\mathbf{x}_n)\right) - N \log Z(\boldsymbol{\lambda}^{\text{MP}}) \rightarrow \sum_n \left(\boldsymbol{\lambda}^{\text{MP}T} \mathbf{f}(\mathbf{t}_n, \mathbf{x}_n) - \log Z(\boldsymbol{\lambda}^{\text{MP}}, \mathbf{x}_n)\right) \tag{12}$$

where

$$C_{\mathbf{x}_n} = \mathbb{E}[\mathbf{f}(\mathbf{t}, \mathbf{x}_n)\mathbf{f}(\mathbf{t}, \mathbf{x}_n)^T]_{p(\mathbf{t}|\mathbf{x}_n)} - \mathbb{E}[\mathbf{f}(\mathbf{t}, \mathbf{x}_n)]_{p(\mathbf{t}|\mathbf{x}_n)}\mathbb{E}[\mathbf{f}(\mathbf{t}, \mathbf{x}_n)]_{p(\mathbf{t}|\mathbf{x}_n)}^T \tag{13}$$

and where all averages are taken over distributions $p(\mathbf{t}|\mathbf{x}_n, \boldsymbol{\lambda}^{\text{MP}})$ at the MAP value $\boldsymbol{\lambda}^{\text{MP}}$ of the conditional log-likelihood $\sum_n \log p(\mathbf{t}_n|\mathbf{x}_n, \boldsymbol{\lambda})$.

## 4 Experiments

In the following experiments we probe the accuracy of the Bethe-Laplace(BP-LR) approximation. In these experiments we have focussed on comparing the value of the estimated log marginal likelihood with "annealed importance sampling" (AIS), which we treat as ground truth[9, 1]. We have focussed on this performance measure because the marginal likelihood is the relevant quantity for both Bayesian model averaging as well as model selection.

We perform experiments on synthetic data as well as a real-world dataset. For the synthetic data, we use Boltzman machine models (binary undirected graphical models with pairwise interactions) because we believe that the results will be representative of multi-state models and because the implementation of the linear response approximation is straightforward in this case (see appendix A).

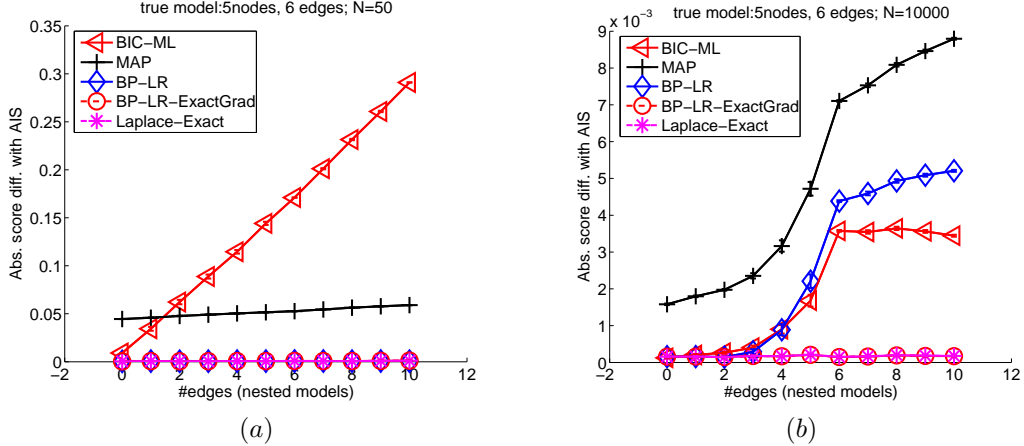

Figure 2: Mean difference in scores with AIS (synthetic data). Error-bars are too small to see.

Scores computed using the proposed method (BP-LR) were compared against MAP scores (or penalized log-likelihood) where we retain only the first three terms in equation (8) and the commonly used BIC-ML scores where we ignore all $O(1)$ terms (i.e retain only terms 1, 2 and 5). BIC-ML uses the maximum likelihood value $\boldsymbol{\lambda}^{\text{ML}}$ instead of $\boldsymbol{\lambda}^{\text{MP}}$. We also evaluate two other scores - BP-LR-ExactGrad where we use exact gradients to compute the $\boldsymbol{\lambda}^{\text{MP}}$ and Laplace-Exact which is same as BP-LR-ExactGrad but with C computed exactly as well. Note that these last two methods are practical only for models with small tree-width. Nevertheless they are useful here to illustrate the effect of the bias from BP.

## 4.1 Synthetic Data

We generated $50$ different random structures on 5 nodes. For each we sample 6 different sets of parameters with weights $w \sim \mathcal{U}\{[-d, -d+\epsilon] \cup [d, d+\epsilon]\}, d > 0, \epsilon = \frac{0.1}{4}$ and biases $b \sim \mathcal{U}[-1, 1]$ and varying the edge strength $d$ in $[\frac{0.1}{4}, \frac{0.2}{4}, \frac{0.5}{4}, \frac{1.0}{4}, \frac{1.5}{4}, \frac{2.0}{4}]$. We then generated $N = 10000$ samples from each of these $(50 \times 6)$ models using exact sampling by exhaustive enumeration.

In the first experiment we picked a random dataset/model with $d = \frac{0.5}{4}$ (the true structure had 6 edges) and studied the variation of different scores with model complexity. We define an ordering on models based on complexity by using *nested* model sequences. These are such that a model appearing later in the sequence contains all edges from models appearing earlier. Figure (1) shows the results for two such random nested sequences around the true model, for the number of datacases $N = 50$ and $N = 10000$ respectively. The error-bars for AIS are over 10 parallel annealing runs which we see are very small. We repeated the plots for multiple such model sequences and the results were similar. Figure (2) shows the average absolute difference of each score with the AIS score over 50 sequences. From these one can see that BP-LR is very accurate at low $N$. As known in the literature, BIC-ML tends to over-penalize model complexity. At large $N$, the performance of all methods improve but BP-LR does slightly worse than the BIC-ML.

In order to better understand the performance of various scores with $N$, we took the datasets at $d = \frac{0.5}{4}$ and computed scores at various values of $N$. At each value, we find the absolute difference in the score assigned to the true structure with the corresponding AIS score. These are then averaged over the 50 datasets. The results are shown in figure (3). We note that all BP-LR methods are about two orders of magnitude more accurate than methods that ignore the $\mathcal{O}(1)$ term based on $C$. However, as we increase $N$, BP-LR based on $\boldsymbol{\lambda}^{\text{MP}}$ computed using BP significantly deteriorates. This does not happen with both BP-LR methods based on $\boldsymbol{\lambda}^{\text{MP}}$ computed using exact gradients (i.e. BP-LR-ExactGrad and Laplace-Exact). Since the latter two methods perform identically, we conclude that it is not the approximation of $C$ by linear response that breaks down, but rather that the bias in $\boldsymbol{\lambda}^{\text{MP}}$ is the reason that the estimate of $C$ becomes unreliable. We conjecture that this happens when the bias becomes of the order of the standard deviation of the posterior distribution. Since the bias is

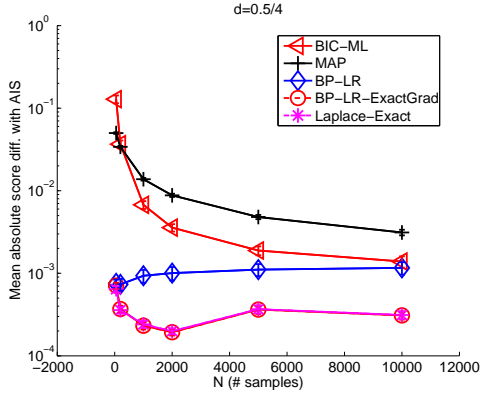
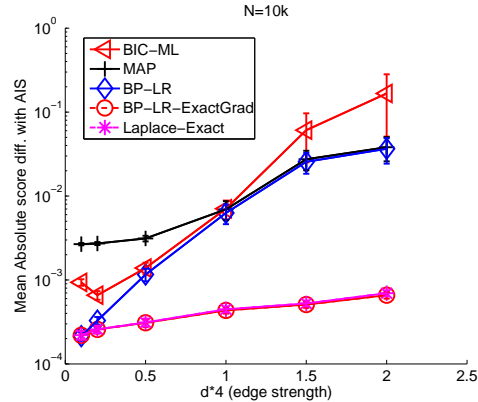

Figure 3: Variation of score accuracy with $N$     Figure 4: Variation of score accuracy with $d$

constant but the variance in the posterior decreases as $\mathcal{O}(1/N)$ this phenomenon is bound to happen for some value of $N$.

Finally since our BP-LR method relies on the BP approximation which is known to break down at strong interactions, we investigated the performance of various scores with $d$. Again at each value of $d$ we compute the average absolute difference in the scores assigned to the true structure by a method and AIS. We use $N = 10000$ to keep the effect of $N$ minimal. Results are shown in figure (4). As expected all BP based methods deteriorate with increasing $d$. The exact methods show that one can improve performance by having a more accurate estimate of $\boldsymbol{\lambda}^{\text{MP}}$.

## 4.2   Real-world Data

To see the performance of the BP-LR on real world data, we implemented a linear chain CRF on the "newsgroup FAQ dataset"[2] [4]. This dataset contains $48$ files where each line can be either a header, a question or an answer. The problem is binarized by only retaining the question/answer lines. For each line we use $24$ binary features $g^a(x) = 0/1$, $a = 1, .., 24$ as provided by [4]. These are used to define state and transition features using: $f_i^a(t_i, x_i) = t_i g^a(x_i)$ and $f_i^a(t_i, t_{i+1}, x_i) = t_i t_{i+1} g^a(x_i)$ where $i$ denotes the line in a document and $a$ indexes the $24$ features.

We generated a random sequence of models by incrementally adding some state features and then some transition features. We then score each model using MAP, BIC-MAP (which is same as BIC-ML but with $\boldsymbol{\lambda}^{\text{MP}}$), AIS and Laplace-Exact. Note that since the graph is a chain, BP-LR is equivalent to BP-LR-ExactGrad and Laplace-Exact. We use $N = 2$ files each truncated to $100$ lines. The results are shown in figure (5). Here again, the Laplace-Exact agrees very closely with AIS compared to the other two methods. (Another less relevant observation is that the scores flatten out around the point where we stop adding the state features showing their importance compared to transition features).

## 5   Discussion

The main conclusion from this study is that the Bethe-Laplace approximation can give an excellent approximation to the marginal likelihood for small datasets. We discovered an interesting phenomenon, namely that as $N$ grows the error in the $\mathcal{O}(1)$ term based on the covariance between features *increases*. We found that this term can give an enormous boost in accuracy for small $N$ (up to two orders of magnitude), but its effect can be detrimental for large $N$. We conjecture that this switch-over point takes place when the bias in $\boldsymbol{\lambda}^{\text{MP}}$ becomes of the order of the standard deviation in the posterior (which decreases as $1/N$). At that point the second derivative of the log-likelihood in the Taylor expansion becomes unreliable.

There are a number of ways to improve the accuracy of approximation. One approach is to use higher order Kikuchi approximations to replace the Bethe approximation. Linear response results are also

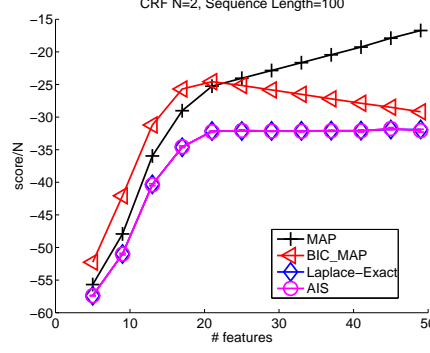

Figure 5: Comparision of various scores on real-world dataset

available for this case [12]. A second improvement could come from improving the estimate of $\boldsymbol{\lambda}^{\text{MP}}$ using alternative learning techniques such as contrastive divergence or alternative sample-based approaches. As discussed above, less bias in $\boldsymbol{\lambda}^{\text{MP}}$ will make the covariance term useful for larger $N$.

Finally, the case of hidden variables needs to be addressed. It is not hard to imagine how to extend the techniques proposed in this paper to hidden variables in theory, but we haven't run the experiments necessary to make claims about its performance. This, we leave for future study.

## A    Computation of $C$ for Boltzman Machines

For binary variables and pairwise interactions we define the variables as $\boldsymbol{\lambda} = \{\theta_i, w_{ij}\}$ where $\theta_i$ is a parameter multiplying the node-feature $x_i$ and $w_{ij}$ the parameter multiplying the edge feature $x_i x_j$. Moreover, we'll define the following independent quantities $q_i = p(x_i = 1)$ and $\xi_{ij} = p(x_i = 1, x_j = 1)$. Note that all other quantities, e.g. $p(x_i = 1, x_j = 0)$ are functions of $\{q_i, \xi_{ij}\}$.

In the following we will assume that $\{q_i, \xi_{ij}\}$ are computed using belief propagation (BP). At the fixed points of BP the following relations hold [14],

$$w_{ij} = \log\left(\frac{\xi_{ij}(\xi_{ij} + 1 - q_i - q_j)}{(q_i - \xi_{ij})(q_j - \xi_{ij})}\right) \qquad \theta_i = \log\left(\frac{(1 - q_i)^{z_i - 1}\prod_{j \in N(i)}(q_i - \xi_{ij})}{q_i^{z_i - 1}\prod_{j \in N(i)}(\xi_{ij} + 1 - q_i - q_j)}\right) \quad (14)$$

where $N(i)$ are neighboring nodes of node $i$ in the graph and $z_i = |N(i)|$ is the number of neighbors of node $i$.

To compute the covariance matrix we first compute its inverse from eqns.14 as follows, $C^{-1} = \begin{bmatrix} \frac{\partial \boldsymbol{\theta}}{\partial \mathbf{q}} & \frac{\partial \boldsymbol{\theta}}{\partial \boldsymbol{\xi}} \\ \frac{\partial \mathbf{w}}{\partial \mathbf{q}} & \frac{\partial \mathbf{w}}{\partial \boldsymbol{\xi}} \end{bmatrix}$ and subsequently take its inverse. The four terms in this matrix are given by,

$$\frac{\partial \theta_i}{\partial q_k} = \left[\frac{1 - z_i}{q_i(1 - q_i)} + \sum_{j \in N(i)}\left(\frac{1}{q_i - \xi_{ij}} + \frac{1}{\xi_{ij} + 1 - q_i - q_j}\right)\right]\delta_{ik} \qquad (15)$$

$$\frac{\partial \theta_i}{\partial \xi_{jk}} = \left[\frac{-1}{q_i - \xi_{ik}} + \frac{1}{\xi_{ik} + 1 - q_i - q_k}\right]\delta_{ij} + \left[\frac{-1}{q_i - \xi_{ij}} + \frac{1}{\xi_{ij} + 1 - q_i - q_j}\right]\delta_{ik} \qquad (16)$$

$$\frac{\partial W_{ij}}{\partial q_k} = \left[\frac{1}{q_i - \xi_{ij}} - \frac{1}{\xi_{ij} + 1 - q_i - q_j}\right]\delta_{ik} + \left[\frac{1}{q_j - \xi_{ij}} - \frac{1}{\xi_{ij} + 1 - q_i - q_j}\right]\delta_{jk} \qquad (17)$$

$$\frac{\partial W_{ij}}{\partial \xi_{kl}} = \left[\frac{1}{\xi_{ij}} + \frac{1}{\xi_{ij} + 1 - q_i - q_j} + \frac{1}{q_i - \xi_{ij}} + \frac{1}{q_j - \xi_{ij}}\right]\delta_{ik}\delta_{jl} \qquad (18)$$

## Acknowledgments

This material is based upon work supported by the National Science Foundation under Grant No. 0447903.

## Footnotes

[1]If one can compute the normalization term exactly (e.g. graphs with small treewidth) or if one can draw perfect samples from the MRF [8](e.g. positive interactions only) then one construct a Markov chain for the posterior.

## References

[1] M.J. Beal and Z. Ghahramani. The variational bayesian EM algorithm for incomplete data: with application to scoring graphical model structures. In *Bayesian Statistics*, pages 453–464. Oxford University Press, 2003.

[2] P. Green and S. Richardson. Hidden markov models and disease mapping. *Journal of the American Statistical Association*, 97(460):1055–1070, 2002.

[3] D. Heckerman. A tutorial on learning with bayesian networks. pages 301–354, 1999.

[4] A. McCallum and D. Freitag F. Pereira. Maximum entropy Markov models for information extraction and segmentation. In *Int'l Conf. on Machine Learning*, pages p.591–598, San Francisco, 2000.

[5] T.S. Jaakkola M.J. Wainwright and A.S. Willsky. Tree-reweighted belief propagation algorithms and approximate ml estimation via pseudo-moment matching. In *AISTATS*, 2003.

[6] J. Møller, A. Pettitt, K. Berthelsen, and R. Reeves. An efficient Markov chain Monte Carlo method for distributions with intractable normalisation constants. *Biometrica*, 93, 2006. to appear.

[7] I. Murray and Z. Ghahramani. Bayesian learning in undirected graphical models: approximate MCMC algorithms. In *Proceedings of the 14th Annual Conference on Uncertainty in Artificial Intelligence (UAI-04)*, San Francisco, CA, 2004.

[8] I. Murray, Z. Ghahramani, and D.J.C. MacKay. Mcmc for doubly-intractable distributions. In *Proceedings of the 14th Annual Conference on Uncertainty in Artificial Intelligence (UAI-06)*, Pittsburgh, PA, 2006.

[9] R.M. Neal. Annealed importance sampling. In *Statistics and Computing*, pages 125–139, 2001.

[10] S. Parise and M. Welling. Learning in markov random fields: An empirical study. In *Proc. of the Joint Statistical Meeting – JSM2005*, 2005.

[11] Y. Qi, M. Szummer, and T.P. Minka. Bayesian conditional random fields. In *Artificial Intelligence and Statistics*, 2005.

[12] K. Tanaka. Probabilistic inference by means of cluster variation method and linear response theory. *IEICE Transactions in Information and Systems*, E86-D(7):1228–1242, 2003.

[13] M. Welling and S. Parise. Bayesian random fields: The Bethe-Laplace approximation. In *UAI*, 2006.

[14] M. Welling and Y.W. Teh. Approximate inference in boltzmann machines. *Artificial Intelligence*, 143:19–50, 2003.

[15] M. Welling and Y.W. Teh. Linear response algorithms for approximate inference in graphical models. *Neural Computation*, 16 (1):197–221, 2004.

[16] J.S. Yedidia, W. Freeman, and Y. Weiss. Constructing free energy approximations and generalized belief propagation algorithms. Technical report, MERL, 2002. Technical Report TR-2002-35.
